# A MCMC approach to Hierarchical Mixture Modelling

**Christopher K. I. Williams**
Institute for Adaptive and Neural Computation
Division of Informatics, University of Edinburgh
5 Forrest Hill, Edinburgh EH1 2QL, Scotland, UK
ckiw@dai.ed.ac.uk          http://anc.ed.ac.uk

## Abstract

There are many hierarchical clustering algorithms available, but these lack a firm statistical basis. Here we set up a hierarchical probabilistic mixture model, where data is generated in a hierarchical tree-structured manner. Markov chain Monte Carlo (MCMC) methods are demonstrated which can be used to sample from the posterior distribution over trees containing variable numbers of hidden units.

## 1 Introduction

Over the past decade or two mixture models have become a popular approach to clustering or competitive learning problems. They have the advantage of having a well-defined objective function and fit in with the general trend of viewing neural network problems in a statistical framework. However, one disadvantage is that they produce a "flat" cluster structure rather than the hierarchical tree structure that is returned by some clustering algorithms such as the agglomerative single-link method (see e.g. [12]). In this paper I formulate a hierarchical mixture model, which retains the advantages of the statistical framework, but also features a tree-structured hierarchy.

The basic idea is illustrated in Figure 1(a). At the root of the tree (level 1) we have a single centre (marked with a ×). This is the mean of a Gaussian with large variance (represented by the large circle). A random number of centres (in this case 3) are sampled from the level 1 Gaussian, to produce 3 new centres (marked with o's). The variance associated with the level 2 Gaussians is smaller. A number of level 3 units are produced and associated with the level 2 Gaussians. The centre of each level 3 unit (marked with a +) is sampled from its parent Gaussian. This hierarchical process could be continued indefinitely, but in this example we generate data from the level 3 Gaussians, as shown by the dots in Figure 1(a).

A three-level version of this model would be a standard mixture model with a Gaussian prior on where the centres are located. In the four-level model the third level centres are clumped together around the second level means, and it is this that distinguishes the model from a flat mixture model. Another view of the generative process is given in Figure 1(b), where the tree structure denotes which nodes are children of particular parents. Note also that this is a *directed* acyclic graph, with the arrows denoting dependence of the position of the child on that of the parent.

In section 2 we describe the theory of probabilistic hierarchical clustering and give a discussion of related work. Experimental results are described in section 3.

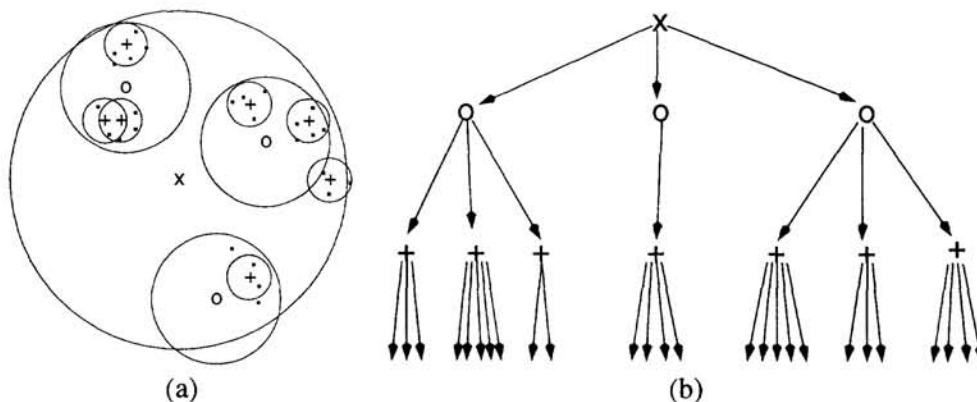

(a)  (b)

Figure 1: The basic idea of the hierarchical mixture model. (a) × denotes the root of the tree, the second level centres are denoted by o's and the third level centres by +'s. Data is generated from the third level centres by sampling random points from Gaussians whose means are the third level centres. (b) The corresponding tree structure.

## 2  Theory

We describe in turn (i) the prior over trees, (ii) the calculation of the likelihood given a data vector, (iii) Markov chain Monte Carlo (MCMC) methods for the inference of the tree structure given data and (iv) related work.

### 2.1  Prior over trees

We describe first the prior over the number of units in each layer, and then the prior on connections between layers. Consider a $L$ layer hierarchical model. The root node is in level 1, there are $n_2$ nodes in level 2, and so on down to $n_L$ nodes on level $L$. These $n$'s are collected together in the vector $\mathbf{n}$. We use a Markovian model for $P(\mathbf{n})$, so that $P(\mathbf{n}) = P(n_1)P(n_2|n_1)\ldots P(n_L|n_{L-1})$ with $P(n_1) = \delta(n_1, 1)$. Currently these are taken to be Poisson distributions offset by 1, so that $P(n_{i+1}|n_i) \sim \mathrm{Po}(\lambda_i n_i) + 1$, where $\lambda_i$ is a parameter associated with level $i$. The offset is used so that there must always be at least one unit in any layer.

Given $\mathbf{n}$, we next consider how the tree is formed. The tree structure describes which node in the $i$th layer is the parent of each node in the $(i + 1)$th layer, for $i = 1, \ldots, L - 1$. Each unit has an indicator vector which stores the index of the parent to which it is attached. We collect all these indicator vectors together into a matrix, denoted $Z(\mathbf{n})$. The probability of a node in layer $(i + 1)$ connecting to any node in layer $i$ is taken to be $1/n_i$. Thus

$$P(\mathbf{n}, Z(\mathbf{n})) = P(\mathbf{n})P(Z(\mathbf{n})|\mathbf{n}) = P(\mathbf{n}) \prod_{i=1}^{L-1} (1/n_i)^{n_{i+1}}.$$

We now describe the generation of a random tree given $\mathbf{n}$ and $Z(\mathbf{n})$. For simplicity we describe the generation of points in 1-d below, although everything can be extended to arbitrary dimension very easily. The mean $\mu^1$ of the level 1 Gaussian is at the origin[1]. The

level 2 means $\mu_j^2$, $j = 1, \ldots, n_2$ are generated from $\mathcal{N}(\mu^1, \sigma_1^2)$, where $\sigma_1^2$ is the variance associated with the level 1 node. Similarly, the position of each level 3 node is generated from its level 2 parent as a displacement from the position of the level 2 parent. This displacement is a Gaussian RV with zero mean and variance $\sigma_2^2$. This process continues on down to the visible variables. In order for this model to be useful, we require that $\sigma_1^2 > \sigma_2^2 > \ldots > \sigma_{L-1}^2$, i.e. that the variability introduced at successive levels declines monotonically (*cf* scaling of wavelet coefficients).

## 2.2 Calculation of the likelihood

The data that we observe are the positions of the points in the final layer; this is denoted **x**. To calculate the likelihood of **x** under this model, we need to integrate out the locations of the means of the hidden variables in levels 2 through to $L - 1$. This can be done explicitly, however, we can shorten this calculation by realizing that given $Z(\mathbf{n})$, the generative distribution for the observables **x** is Gaussian $\mathcal{N}(0, C)$. The covariance matrix $C$ can be calculated as follows. Consider two leaf nodes indexed by $k$ and $l$. The Gaussian RVs that generated the position of these two leaves can be denoted

$$x_k = w_k^1 + w_k^2 + \ldots + w_k^{(L-1)}, \qquad x_l = w_l^1 + w_l^2 + \ldots + w_l^{(L-1)}.$$

To calculate the covariance between $x_k$ and $x_l$, we simply calculate $\langle x_k x_l \rangle$. This depends crucially on how many of the $w$'s are shared between nodes $k$ and $l$ (*cf* path analysis). For example, if $w_k^1 \neq w_l^1$, i.e. the nodes lie in different branches of the tree at level 1, their covariance is zero. If $k = l$, the variance is just the sum of the variances of each RV in the tree. In between, the covariance of $x_k$ and $x_l$ can be determined by finding at what level in the tree their common parent occurs.

Under these assumptions, the log likelihood $L$ of **x** given $Z(\mathbf{n})$ is

$$L = -\frac{1}{2}\mathbf{x}^T C^{-1} \mathbf{x} - \frac{1}{2}\log|C| - \frac{n_L}{2}\log 2\pi. \tag{1}$$

In fact this calculation can be speeded up by taking account of the tree structure (see e.g. [8]). Note also that the posterior means (and variances) of the hidden variables can be calculated based on the covariances between the hidden and visible nodes. Again, this calculation can be carried out more efficiently; see Pearl [11] (section 7.2) for details.

## 2.3 Inference for n and $Z(\mathbf{n})$

Given **n** we have the problem of trying to infer the connectivity structure $Z$ given the observations **x**. Of course what we are interested in is the posterior distribution over $Z$, i.e. $P(Z|\mathbf{x}, \mathbf{n})$. One approach is to use a Markov chain Monte Carlo (MCMC) method to sample from this posterior distribution. A straightforward way to do this is to use the Metropolis algorithm, where we propose changes in the structure by changing the parent of a single node at a time. Note the similarities of this algorithm to the work of Williams and Adams [14] on Dynamic Trees (DTs); the main differences are (i) that disconnections are not allowed, i.e. we maintain a single tree (rather than a forest), and (ii) that the variables in the DT image models are discrete rather than Gaussian.

We also need to consider moves that change **n**. This can be effected with a split/merge move. In the split direction, consider a node with a parent and several children. Split this node and randomly assign the children to the two split nodes. Each of the split nodes keeps the same parent. The probability of accepting this move under the Metropolis-Hastings scheme is

$$\alpha = \min \left(1, \frac{P(\mathbf{n}', Z(\mathbf{n}')|\mathbf{x})Q(\mathbf{n}', Z(\mathbf{n}'); \mathbf{n}, Z(\mathbf{n}))}{P(\mathbf{n}, Z(\mathbf{n})|\mathbf{x})Q(\mathbf{n}, Z(\mathbf{n}); \mathbf{n}', Z(\mathbf{n}'))}\right),$$

where $Q(\mathbf{n}', Z(\mathbf{n}'); \mathbf{n}, Z(\mathbf{n}))$ is the proposal probability of configuration $(\mathbf{n}', Z(\mathbf{n}'))$ given configuration $(\mathbf{n}, Z(\mathbf{n}))$. This scheme is based on the work on MCMC model composition ($MC^3$) by Madigan and York [9], and on Green's work on reversible jump MCMC [5].

Another move that changes $\mathbf{n}$ is to remove "dangling" nodes, i.e. nodes which have no children. This occurs when all the nodes in a given layer "decide" not to use one or more nodes in the layer above.

An alternative to sampling from the posterior is to use approximate inference, such as mean-field methods. These are currently being investigated for DT models [1].

### 2.4 Related work

There are a very large number of papers on hierarchical clustering; in this work we have focussed on expressing hierarchical clustering in terms of probabilistic models. For example Ambros-Ingerson *et al* [2] and Mozer [10] developed models where the idea is to cluster data at a coarse level, subtract out mean and cluster the residuals (recursively). This paper can be seen as a probabilistic interpretation of this idea.

The reconstruction of phylogenetic trees from biological sequence (DNA or protein) information gives rise to the problem of inferring a binary tree from the data. Durbin *et al* [3] (chapter 8) show how a probabilistic formulation of the problem can be developed, and the link to agglomerative hierarchical clustering algorithms as approximations to the full probabilistic method (see §8.6 in [3]). Much of the biological sequence work uses discrete variables, which diverges somewhat from the focus of the current work. However work by Edwards (1970) [4] concerns a branching Brownian-motion process, which has some similarities to the model described above. Important differences are that Edwards' model is in continuous time, and the the variances of the particles are derived from a Wiener process (and so have variance proportional to the lifetime of the particle). This is in contrast to the decreasing sequence of variances at a given number of levels assumed in the above model. One important difference between the model discussed in this paper and the phylogenetic tree model is that points in higher levels of the phylogenetic tree are taken to be individuals at an earlier time in evolutionary history, which is not the interpretation we require here.

An very different notion of hierarchy in mixture models can be found in the work on the AutoClass system [6]. They describe a model involving class hierarchy and inheritance, but their trees specify over which *dimensions* sharing of parameters occurs (e.g. means and covariance matrices for Gaussians). In contrast, the model in this paper creates a hierarchy over examples labelled $1, \ldots, n$ rather than dimensions.

Xu and Pearl [15] discuss the inference of a tree-structured belief network based on knowledge of the covariances of the leaf nodes. This algorithm cannot be applied directly in our case as the covariances are not known, although we note that if multiple runs from a given tree structure were available the covariances might be approximated using sample estimates.

Other ideas concerning hierarchical clustering are discussed in [13] and [7].

## 3 Experiments

We describe two sets of experiments to explore these ideas.

### 3.1 Searching over $Z$ with n fixed

100 4-level random trees were generated from the prior, using values of $\lambda_1 = 1.5$, $\lambda_2 = 2$, $\lambda_3 = 3$, and $\sigma_1^2 = 10$, $\sigma_2^2 = 1$, $\sigma_3^2 = 0.01$. These trees had between 4 and 79 leaf

nodes, with an average of 30. For each tree **n** was kept the same as in the generative tree, and sampling was carried out over $Z$ starting from a random initial configuration. A given node proposes changing its parent, and this proposal is accepted or rejected with the usual Metropolis probability. In one *sweep*, each node in levels 3 and 4 makes such a move. (Level 2 nodes only have one parent so there is no point in such a move there.) To obtain a representative sample of $P(Z(\mathbf{n})|\mathbf{n},\mathbf{x})$, we should run the chain for as long as possible. However, we can also use the chain to find configurations with high posterior probability, and in this case running for longer only increases the chances of finding a better configuration. In our experiments the sampler was run for 100 sweeps. As $P(Z(\mathbf{n})|\mathbf{n})$ is uniform for fixed **n**, the posterior is simply proportional to the likelihood term. It would also be possible to run simulated annealing with the same move set to search explicitly for the *maximum a posteriori* (MAP) configuration.

The results are that for 76 of the 100 cases the tree with the highest posterior probability (HPP) configuration had higher posterior probability than the generative tree, for 20 cases the same tree was found and in 4 cases the HPP solution was inferior to the generative tree. The fact that in almost all cases the sampler found a configuration as good or better than the generative one in a relatively small number of sweeps is very encouraging.

In Figure 2 the generative (left column) and HPP trees for fixed **n** (middle column) are plotted for two examples. In panel (b) note the "dangling" node in level 2, which means that the level 3 nodes to the left end up in a inferior configuration to (a). By contrast, in panel (e) the sampler has found a better (less tangled) configuration than the generative model (d).

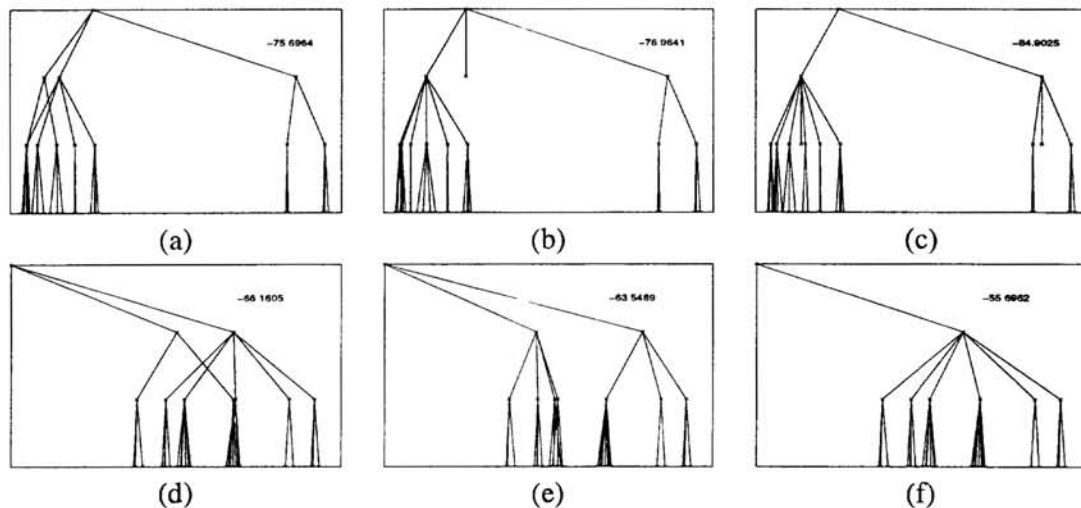

(a)                              (b)                              (c)

(d)                              (e)                              (f)

Figure 2: (a) and (d) show the generative trees for two examples. The corresponding HPP trees for fixed **n** are plotted in (b) and (e) and those for variable **n** in (c) and (f). The number in each panel is the log posterior probability of the configuration. The nodes in levels 2 and 3 are shown located at their posterior means. Apparent non-tree structures are caused by two nodes being plotted almost on top of each other.

## 3.2 Searching over both n and $Z$

Given some data **x** we will not usually know appropriate numbers of hidden units. This motivates searching over both $Z$ and **n**, which can be achieved using the split/merge moves discussed in section 2.3.

In the experiments below the initial numbers of units in levels 2 and 3 (denoted $\hat{n}_2$ and

$\hat{n}_3$) were set using the simple-minded formulae $\hat{n}_3 = \lceil \dim(\mathbf{x})/\lambda_3 \rceil \ \hat{n}_2 = \lceil \hat{n}_3/\lambda_2 \rceil$. A proper inferential calculation for $n_2$ and $n_3$ can be carried out, but it requires the solution of a non-linear optimization problem. Given $\hat{n}_2$ and $\hat{n}_3$, the initial connection configuration was chosen randomly.

The search method used was to propose a split/merge move (with probability 0.5:0.5) in level 2, then to sample the level 2 to level 3 connections, and then to propose a split-merge move in level 3, and then update the level 3 to level 4 connections. This comprised a single *sweep*, and as above 100 sweeps were used.

Experiments were conducted on the same trees used in section 3.1. In this case the results were that for 50 out of the 100 cases, the HPP configuration had higher posterior probability than the generative tree, for 11 cases the same tree was found and in 39 cases the HPP solution was inferior to the generative tree. Overall these results are less good than the ones in section 3.1, but it should be remembered that the search space is now much larger, and so it would be expected that one would need to search longer. Comparing the results from fixed $\mathbf{n}$ against those with variable $\mathbf{n}$ shows that in 42 out of 100 cases the variable $\mathbf{n}$ method gave a higher posterior probability, in 45 cases it was lower and in 13 cases the same trees were found.

The rightmost column of Figure 2 shows the HPP configurations when sampling with variable $\mathbf{n}$ on the two examples discussed above. In panel (c) the solution found is not very dissimilar to that in panel (b), although the overall probability is lower. In (f), the solution found uses just one level 2 centre rather than two, and obtains a higher posterior probability than the configurations in (e) and (d).

# 4 Discussion

The results above indicate that the proposed model behaves sensibly, and that reasonable solutions can be found with relatively short amounts of search. The method has been demonstrated on univariate data, but extending it to multivariate Gaussian data for which each dimension is independent given the tree structure is very easy as the likelihood calculation is independent on each dimension.

There are many other directions is which the model can be developed. Firstly, the model as presented has uniform mixing proportions, so that children are equally likely to connect to each potential parent. This can be generalized so that there is a non-uniform vector of connection probabilities in each layer. Also, given a tree structure and independent Dirichlet priors over these probability vectors, these parameters can be integrated out analytically. Secondly, the model can be made to generate iid data by regarding the penultimate layer as mixture centres; in this case the term $P(n_L|n_{L-1})$ would be ignored when computing the probability of the tree. Thirdly, it would be possible to add the variance variables to the MCMC scheme, e.g. using the Metropolis algorithm, after defining a suitable prior on the sequence of variances $\sigma_1^2, \ldots, \sigma_{L-1}^2$. The constraint that all variances in the same level are equal could also be relaxed by allowing them to depend on hyperparameters set at every level. Fourthly, there may be improved MCMC schemes that can be devised. For example, in the current implementation the posterior means of the candidate units are not taken into account when proposing merge moves (*cf* [5]). Fifthly, for the multivariate Gaussian version we can consider a tree-structured factor analysis model, so that higher levels in the tree need not have the same dimensionality as the data vectors.

One can also consider a version where each dimension is a multinomial rather than a continuous variable. In this case one might consider a model where a multinomial parameter vector $\boldsymbol{\theta}_l$ in the tree is generated from its parent by $\boldsymbol{\theta}_l = \gamma \boldsymbol{\theta}_{l-1} + (1 - \gamma)\mathbf{r}$ where $\gamma \in [0, 1]$ and $\mathbf{r}$ is a random vector of probabilities. An alternative model could be to build a tree

structured prior on the $\alpha$ parameters of the Dirichlet prior for the multinomial distribution.

**Acknowledgments**

This work is partially supported through EPSRC grant GR/L78161 *Probabilistic Models for Sequences*. I thank the Gatsby Computational Neuroscience Unit (UCL) for organizing the "Mixtures Day" in March 1999 and supporting my attendance, and Peter Green, Phil Dawid and Peter Dayan for helpful discussions at the meeting. I also thank Amos Storkey for helpful discussions and Magnus Rattray for (accidentally!) pointing me towards the chapters on phylogenetic trees in [3].

## Footnotes

[1]It is easy to relax this assumption so that $\mu^1$ has a prior Gaussian distribution, or is located at some point other than the origin.

# References

[1] N. J. Adams, A. Storkey, Z. Ghahramani, and C. K. I. Williams. MFDTs: Mean Field Dynamic Trees. Submitted to ICPR 2000, 1999.

[2] J. Ambros-Ingerson, R. Granger, and G. Lynch. Simulation of Paleocortex Performs Hierarchical Clustering. *Science*, 247:1344–1348, 1990.

[3] R. Durbin, S. Eddy, A. Krogh, and G. Mitchison. *Biological Sequence Analysis*. Cambridge University Press, Cambridge, UK, 1998.

[4] A. W. F. Edwards. Estimation of the Branch Points of a Branching Diffusion Process. *Journal of the Royal Statistical Society B*, 32(2):155–174, 1970.

[5] P. J. Green. Reversible Jump Markov chain Monte Carlo computation and Bayesian model determination. *Biometrika*, 82(4):711–732, 1995.

[6] R. Hanson, J. Stutz, and P. Cheeseman. Bayesian Classification with Correlation and Inheritance. In *IJCAI-91: Proceedings of the Twelfth International Joint Conference on Artificial Intelligence*, 1991. Sydney, Australia.

[7] T. Hofmann and J. M. Buhmann. Hierarchical Pairwise Data Clustering by Mean-Field Annealing. In F. Fogelman-Soulie and P. Gallinari, editors, *Proc. ICANN 95*. EC2 et Cie, 1995.

[8] M. R. Luettgen and A. S. Willsky. Likelihood Calculation for a Class of Multiscale Stochastic Models, with Application to Texture Discrimination. *IEEE Trans. Image Processing*, 4(2):194–207, 1995.

[9] D. Madigan and J. York. Bayesian Graphical Models for Discrete Data. *International Statistical Review*, 63:215–232, 1995.

[10] M. C. Mozer. Discovering Discrete Distributed Representations with Iterated Competitive Learning. In R. P. Lippmann, J. E. Moody, and D. S. Touretzky, editors, *Advances in Neural Information Processing Systems 3*. Morgan Kaufmann, 1991.

[11] J. Pearl. *Probabilistic Reasoning in Intelligent Systems: Networks of Plausible Inference*. Morgan Kaufmann, San Mateo, CA, 1988.

[12] B. Ripley. *Pattern Recognition and Neural Networks*. Cambridge University Press, Cambridge, UK, 1996.

[13] N. Vasconcelos and A. Lippmann. Learning Mixture Hierarchies. In M. S. Kearns, S. A. Solla, and D. A. Cohn, editors, *Advances in Neural Information Processing Systems 11*, pages 606–612. MIT Press, 1999.

[14] C. K. I. Williams and N. J. Adams. DTs: Dynamic Trees. In M. J. Kearns, S. A. Solla, and D. A. Cohn, editors, *Advances in Neural Information Processing Systems 11*. MIT Press, 1999.

[15] L. Xu and J. Pearl. Structuring Causal Tree Models with Continuous Variables. In L. N. Kanal, T. S. Levitt, and J. F. Lemmer, editors, *Uncertainty in Artificial Intelligence 3*. Elsevier, 1989.